# Spike Timing-Dependent Plasticity in the Address Domain

**R. Jacob Vogelstein**[1], **Francesco Tenore**[2], **Ralf Philipp**[2], **Miriam S. Adlerstein**[2],
**David H. Goldberg**[2] **and Gert Cauwenberghs**[2]
[1]Department of Biomedical Engineering
[2]Department of Electrical and Computer Engineering
Johns Hopkins University, Baltimore, MD 21218
{*jvogelst,fra,rphilipp,mir,goldberg,gert*}*@jhu.edu*

## Abstract

Address-event representation (AER), originally proposed as a means to communicate sparse neural events between neuromorphic chips, has proven efficient in implementing large-scale networks with arbitrary, configurable synaptic connectivity. In this work, we further extend the functionality of AER to implement arbitrary, configurable synaptic plasticity in the address domain. As proof of concept, we implement a biologically inspired form of spike timing-dependent plasticity (STDP) based on relative timing of events in an AER framework. Experimental results from an analog VLSI integrate-and-fire network demonstrate address domain learning in a task that requires neurons to group correlated inputs.

## 1 Introduction

It has been suggested that the brain's impressive functionality results from massively parallel processing using simple and efficient computational elements [1]. Developments in neuromorphic engineering and address-event representation (AER) have provided an infrastructure suitable for emulating large-scale neural systems in silicon, *e.g.,* [2, 3]. Although an integral part of neuromorphic engineering since its inception [1], only recently have implemented systems begun to incorporate adaptation and learning with biological models of synaptic plasticity.

A variety of learning rules have been realized in neuromorphic hardware [4, 5]. These systems usually employ circuitry incorporated into the individual cells, imposing constraints on the nature of inputs and outputs of the implemented algorithm. While well-suited to small assemblies of neurons, these architectures are not easily scalable to networks of hundreds or thousands of neurons. Algorithms based both on continuous-valued "intracellular" signals and discrete spiking events have been realized in this way, and while analog computations may be performed better at the cellular level, we argue that it is advantageous to implement spike-based learning rules in the address domain. AER-based systems are inherently scalable, and because the encoding and decoding of events is performed at the periphery, learning algorithms can be arbitrarily complex without increasing the size of repeating neural units. Furthermore, AER makes no assumptions about the signals repre-

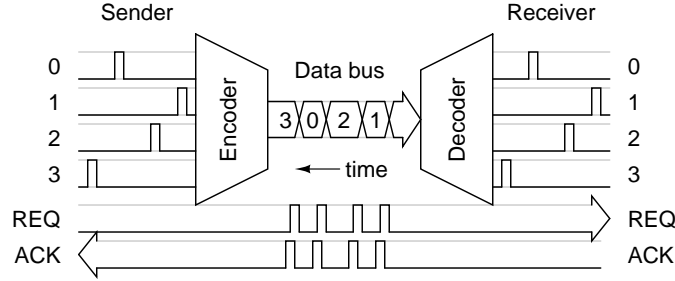

Figure 1: Address-event representation. Sender events are encoded into an address, sent over the bus, and decoded. Handshaking signals REQ and ACK are required to ensure that only one cell pair is communicating at a time. Note that the time axis goes from right to left.

sented as spikes, so learning can address any measure of cellular activity. This flexibility can be exploited to achieve learning mechanisms with high degrees of biological realism.

Much previous work has focused on rate-based Hebbian learning (*e.g.,* [6]), but recently, the possibility of modifying synapses based on the timing of action potentials has been explored in both the neuroscience [7, 8] and neuromorphic engineering disciplines [9]–[11]. This latter hypothesis gives rise to the possibility of learning based on causality, as opposed to mere correlation. We propose that AER-based neuromorphic systems are ideally suited to implement learning rules founded on this notion of spike-timing dependent plasticity (STDP). In the following sections, we describe an implementation of one biologically-plausible STDP learning rule and demonstrate that table-based synaptic connectivity can be extended to table-based synaptic plasticity in a scalable and reconfigurable neuromorphic AER architecture.

## 2    Address-domain architecture

Address-event representation is a communication protocol that uses time-multiplexing to emulate extensive connectivity [12] (Fig. 1). In an AER system, one array of neurons encodes its activity in the form of spikes that are transmitted to another array of neurons. The "brute force" approach to communicating these signals would be to use one wire for each pair of neurons, requiring $N$ wires for $N$ cell pairs. However, an AER system identifies the location of a spiking cell and encodes this as an address, which is then sent across a shared data bus. The receiving array decodes the address and routes it to the appropriate cell, reconstructing the sender's activity. Handshaking signals REQ and ACK are required to ensure that only one cell pair is using the data bus at a time. This scheme reduces the required number of wires from $N$ to $\sim \log_2 N$. Two pieces of information uniquely identify a spike: its location, which is explicitly encoded as an address, and the time that it occurs, which need not be explicitly encoded because the events are communicated in real-time. The encoded spike is called an *address-event*.

In its original formulation, AER implements a one-to-one connection topology, which is appropriate for emulating the optic and auditory nerves [12, 13]. To create more complex neural circuits, convergent and divergent connectivity is required. Several authors have discussed and implemented methods of enhancing the connectivity of AER systems to this end [14]–[16]. These methods call for a memory-based projective field mapping that enables routing an address-event to multiple receiver locations.

The enhanced AER system employed in this paper is based on that of [17], which en-

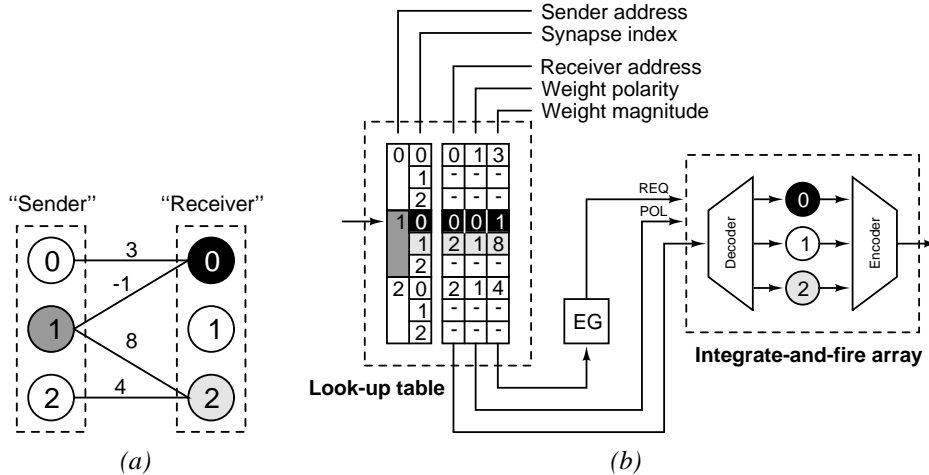

Figure 2: Enhanced AER for implementing complex neural networks. *(a)* Example neural network. The connections are labeled with their weight values. *(b)* The network in *(a)* is mapped to the AER framework by means of a look-up table.

ables continuous-valued synaptic weights by means of graded (probabilistic or deterministic) transmission of address-events. This architecture employs a look-up table (LUT), an integrate-and-fire address-event transceiver (IFAT), and some additional support circuitry. Fig. 2 shows how an example two-layer network can be mapped to the AER framework. Each row in the table corresponds to a single synaptic connection—it contains information about the sender location, the receiver location, the connection polarity (excitatory or inhibitory), and the connection magnitude. When a spike is sent to the system, the sender address is used as an index into the LUT and a signal activates the event generator (EG) circuit. The EG scrolls through all the table entries corresponding to synaptic connections from the sending neuron. For each synapse, the receiver address and the spike polarity are sent to the IFAT, and the EG initiates as many spikes as are specified in the weight magnitude field.

Events received by the IFAT are temporally and spatially integrated by analog circuitry. Each integrate-and-fire cell receives excitatory and inhibitory inputs that increment or decrement the potential stored on an internal capacitance. When this potential exceeds a given threshold, the cell generates an output event and broadcasts its address to the AE arbiter. The physical location of neurons in the array is inconsequential as connections are routed through the LUT, which is implemented in random-access memory (RAM) outside of the chip.

An interesting feature of the IFAT is that it is insensitive to the *timescale* over which events occur. Because internal potentials are not subject to decay, the cells' activities are only sensitive to the *order* of the events. Effects of leakage current in real neurons are emulated by regularly sending inhibitory events to all of the cells in the array. Modulating the timing of the "global decay events" allows us to dynamically warp the time axis.

We have designed and implemented a prototype system that uses the IFAT infrastructure to implement massively connected, reconfigurable neural networks. An example setup is described in detail in [17], and is illustrated in Fig. 3. It consists of a custom VLSI IFAT chip with a $1024$-neuron array, a RAM that stores the look-up table, and a microcontroller unit (MCU) that realizes the event generator.

As discussed in [18, p. 91], a synaptic weight $w$ can be expressed as the combined effect

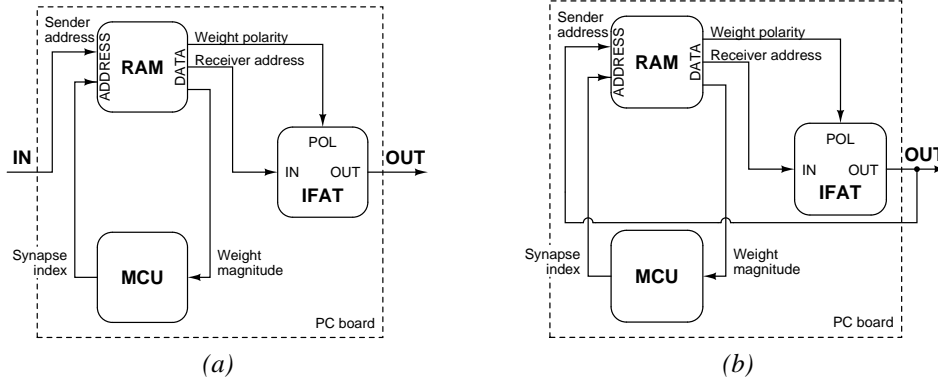

Figure 3: Hardware implementation of enhanced AER. The elements are an integrate-and-fire array transceiver (IFAT) chip, a random-access memory (RAM) look-up table, and a microcontroller unit (MCU). *(a)* Feedforward mode. Input events are routed by the RAM look-up table, and integrated by the IFAT chip. *(b)* Recurrent mode. Events emitted by the IFAT are sent to the look-up table, where they are routed back to the IFAT. This makes virtual connections between IFAT cells.

of three physical mechanisms:

$$w = npq \tag{1}$$

where $n$ is the number of quantal neurotransmitter sites, $p$ is the probability of synaptic release per site, and $q$ is the measure of the postsynaptic effect of the synapse. Many early neural network models held $n$ and $p$ constant and attributed all of the variability in the weight to $q$. Our architecture is capable of varying all three components: $n$ by sending multiple events to the same receiver location, $p$ by probabilistically routing the events (as in [17]), and $q$ by varying the size of the potential increments and decrements in the IFAT cells. In the experiments described in this paper, the transmission of address-events is deterministic, and the weight is controlled by varying the number of events per synapse, corresponding to a variation in $n$.

## 3   Address-domain learning

The AER architecture lends itself to implementations of synaptic plasticity, since information about presynaptic and postsynaptic activity is readily available and the contents of the synaptic weight fields in RAM are easily modifiable "on the fly." As in biological systems, synapses can be dynamically created and pruned by inserting or deleting entries in the LUT.

Like address domain connectivity, the advantage of address domain plasticity is that the constituents of the implemented learning rule are not constrained to be local in space or time. Various forms of learning algorithms can be mapped onto the same architecture by reconfiguring the MCU interfacing the IFAT and the LUT.

Basic forms of Hebbian learning can be implemented with no overhead in the address domain. When a presynaptic event, routed by the LUT through the IFAT, elicits a postsynaptic event, the synaptic strength between the two neurons is simply updated by incrementing the data field of the LUT entry at the active address location. A similar strategy can be adopted for other learning rules of the incremental outer-product type, such as delta-rule or back-propagation supervised learning.

Non-local learning rules require control of the LUT address space to implement spatial and/or temporal dependencies. Most interesting from a biological perspective are forms of

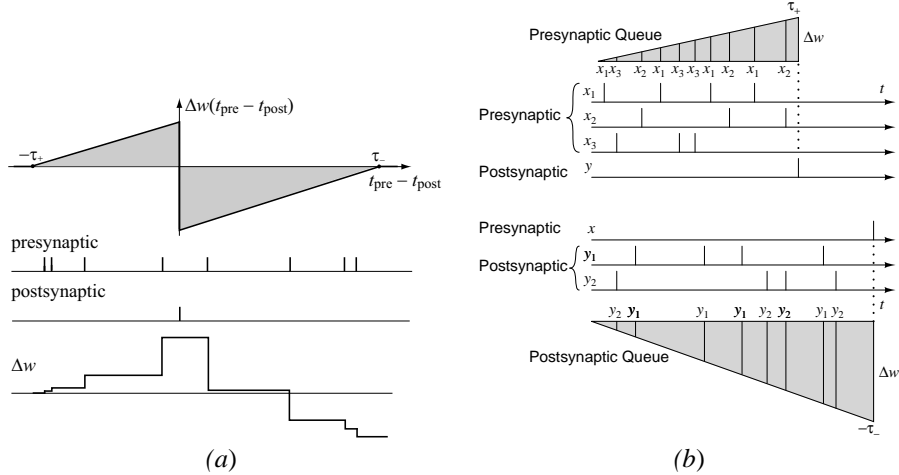

<div align="center">

*(a)*                                            *(b)*

</div>

Figure 4: Spike timing-dependent plasticity (STDP) in the address domain. *(a)* Synaptic updates $\Delta w$ as a function of the relative timing of presynaptic and postsynaptic events, with asymmetric windows of anti-causal and causal regimes $\tau_- > \tau_+$. *(b)* Address-domain implementation using presynaptic (top) and postsynaptic (bottom) event queues of window lengths $\tau_+$ and $\tau_-$.

spike timing-dependent plasticity (STDP).

## 4   Spike timing-dependent plasticity

Learning rules based on STDP specify changes in synaptic strength depending on the time interval between each pair of presynaptic and postsynaptic events. "Causal" postsynaptic events that succeed presynaptic action potentials (APs) by a short duration of time potentiate the synaptic strength, while "anti-causal" presynaptic events succeeding postsynaptic APs by a short duration depress the synaptic strength. The amount of strengthening or weakening is dependent on the exact time of the event within the causal or anti-causal regime, as illustrated in Fig. 4 (a). The weight update has the form

$$\Delta w = \begin{cases} -\eta[\tau_- - (t_{\mathrm{pre}} - t_{\mathrm{post}})] & 0 \le t_{\mathrm{pre}} - t_{\mathrm{post}} \le \tau_- \\ \eta[\tau_+ + (t_{\mathrm{pre}} - t_{\mathrm{post}})] & -\tau_+ \le t_{\mathrm{pre}} - t_{\mathrm{post}} \le 0 \\ 0 & \text{otherwise} \end{cases} \tag{2}$$

where $t_{\mathrm{pre}}$ and $t_{\mathrm{post}}$ denote time stamps of presynaptic and postsynaptic events.

For stable learning, the time windows of causal and anti-causal regimes $\tau_+$ and $\tau_-$ are subject to the constraint $\tau_+ < \tau_-$. For more general functional forms of STDP $\Delta w(t_{\mathrm{pre}} - t_{\mathrm{post}})$, the area under the synaptic modification curve in the anti-causal regime must be greater than that in the causal regime to ensure convergence of the synaptic strengths [7].

The STDP synaptic modification rule (2) is implemented in the address domain by augmenting the AER architecture with two event queues, one each for presynaptic and postsynaptic events, shown in Figure 4 (b). Each time a presynaptic event is generated, the sender's address is entered into a queue with an associated value of $\tau_+$. All values in the queue are decremented every time a global decay event is observed, marking one unit of time $T$. A postsynaptic event triggers a sequence of synaptic updates by iterating backwards through the queue to find the causal spikes, in turn locating the synaptic strength entries in the LUT corresponding to the sender addresses and synaptic index, and increasing

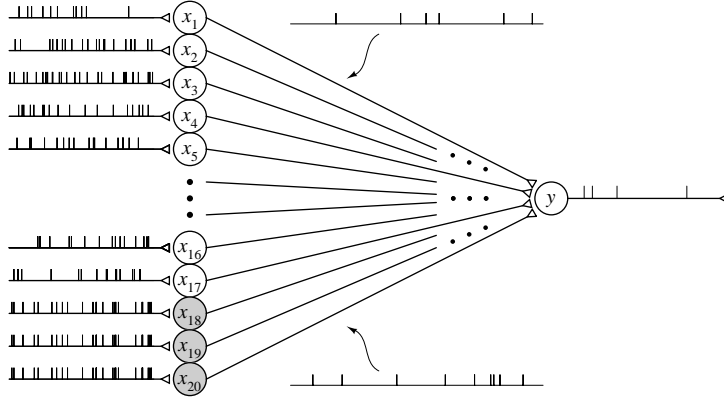

Figure 5: Pictorial representation of our experimental neural network, with actual spike train data sent from the workstation to the first layer. All cells are identical, but $x_{18} \ldots x_{20}$ (shaded) receive correlated inputs. Activity becomes more sparse in the hidden and output layers as the IFAT integrates spatiotemporally. Note that connections are virtual, specified in the RAM look-up-table.

the synaptic strengths in the LUT according to the values stored in the queue. Anti-causal events require an equivalent set of operations, matching each incoming presynaptic spike with a second queue of postsynaptic events. In this case, entries in the queue are initialized with a value of $\tau_-$ and decremented after every interval of time $T$ between decay events, corresponding to the decrease in strength to be applied at the presynaptic/postsynaptic pair.

We have chosen a particularly simple form of the synaptic modification function (2) as proof of principle in the experiments. More general functions can be implemented by a table that maps time bins in the history of the queue to specified values of $\Delta w(nT)$, with positive values of $n$ indexing the postsynaptic queue, and negative values indexing the presynaptic queue.

## 5   Experimental results

We have implemented a Hebbian spike timing-based learning rule on a network of 21 neurons using the IFAT system (Fig. 5). Each of the 20 neurons in the input layer is driven by an externally supplied, randomly generated list of events. Sufficiently high levels of input cause these neurons to produce spikes that subsequently drive the output layer. All events are communicated over the address-event bus and are monitored by a workstation communicating with the MCU and RAM. As shown in [7], temporally asymmetric Hebbian learning using STDP is useful for detecting correlations between inputs. We have proved that this can be accomplished in hardware in the address domain by presenting the network with stimulus patterns containing a set of correlated inputs and a set of uncorrelated inputs: neurons $x_1 \ldots x_{17}$ are all stimulated independently with a probability of 0.05 per unit of time, while neurons $x_{18} \ldots x_{20}$ have the same likelihood of stimulation but are always activated together. Thus, over a sufficiently long period of time each neuron in the input layer will receive the same amount of activation, but the correlated group will fire synchronous spikes more frequently than any other combination of neurons.

In the implemented learning rule (2), causal activity results in synaptic strengthening and anti-causal activity results in synaptic weakening. As described in Section 4, for an anti-causal regime $\tau_-$ larger than the causal regime $\tau_+$, random activity results in overall weak-

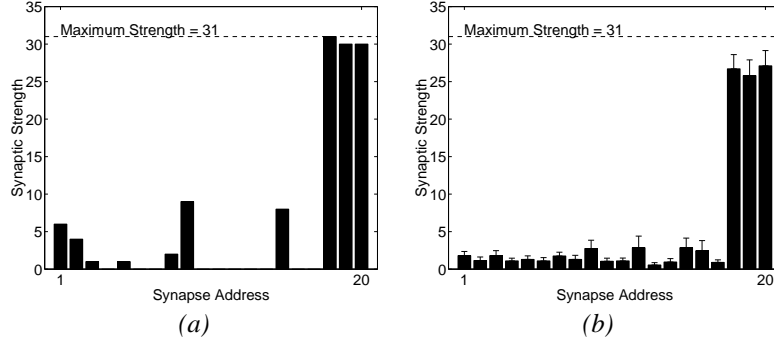

Figure 6: Experimental synaptic strengths in the second layer, recorded from the IFAT system after the presentation of 200,000 input events. *(a)* Typical experimental run. *(b)* Average (+SE) over 20 experimental runs.

ening of a synapse. All synapses connecting the input and output layers are equally likely to be active during an anti-causal regime. However, the increase in average contribution to the postsynaptic membrane potential for the correlated group of neurons renders this population slightly more likely to be active during the causal regime than any single member of the uncorrelated group. Therefore, the synaptic strengths for this group of neurons will increase with respect to the uncorrelated group, further augmenting their likelihood of causing a postsynaptic spike. Over time, this positive feedback results in a random but stable distribution of synaptic strengths in which the correlated neurons' synapses form the strongest connections and the remaining neurons are distributed around an equilibrium value for weak connections.

In the experiments, we have chosen $\tau_+ = 3$ and $\tau_- = 6$. An example of a typical distribution of synaptic strengths recorded after 200,000 events have been processed by the input layer is shown in Fig. 6 (a). For the data shown, synapses driving the input layer were fixed at the maximum strength (+31), the rate of decay was −4 per unit of time, and the plastic synapses between the input and output layers were all initialized to +8. Because the events sent from the workstation to the input layer are randomly generated, fluctuations in the strengths of individual synapses occur consistently throughout the operation of the system. Thus, the final distribution of synaptic weights is different each time, but a pattern can be clearly discerned from the average value of synaptic weights after 20 separate trials of 200,000 events each, as shown in Fig. 6 (b).

The system is robust to changes in various parameters of the spike timing-based learning algorithm as well as to modifications in the number of correlated, uncorrelated, and total neurons (data not shown). It also converges to a similar distribution regardless of the initial values of the synaptic strengths (with the constraint that the net activity must be larger than the rate of decay of the voltage stored on the membrane capacitance of the output neuron).

## 6   Conclusion

We have demonstrated that the address domain provides an efficient representation to implement synaptic plasticity that depends on the relative timing of events. Unlike dedicated hardware implementations of learning functions embedded into the connectivity, the address domain implementation allows for learning rules with interactions that are not constrained in space and time. Experimental results verified this for temporally-antisymmetric Hebbian learning, but the framework can be extended to general learning rules, including reward-based schemes [10].

The IFAT architecture can be augmented to include sensory input, physical nearest-neighbor connectivity between neurons, and more realistic biological models of neural computation. Additionally, integrating the RAM and IFAT into a single chip will allow for increased computational bandwidth. Unlike a purely digital implementation or software emulation, the AER framework preserves the continuous nature of the timing of events.

## References

[1] C. Mead, *Analog VLSI and Neural Systems*. Reading, Massachusetts: Addison-Wesley, 1989.

[2] S. R. Deiss, R. J. Douglas, and A. M. Whatley, "A pulse-coded communications infrastructure for neuromorphic systems," in *Pulsed Neural Networks* (W. Maas and C. M. Bishop, eds.), pp. 157–178, Cambridge, MA: MIT Press, 1999.

[3] K. Boahen, "A retinomorphic chip with parallel pathways: Encoding INCREASING, ON, DECREASING, and OFF visual signals," *Analog Integrated Circuits and Signal Processing*, vol. 30, pp. 121–135, February 2002.

[4] G. Cauwenberghs and M. A. Bayoumi, eds., *Learning on Silicon: Adaptive VLSI Neural Systems*. Norwell, MA: Kluwer Academic, 1999.

[5] M. A. Jabri, R. J. Coggins, and B. G. Flower, *Adaptive analog VLSI neural systems*. London: Chapman & Hall, 1996.

[6] T. J. Sejnowski, "Storing covariance with nonlinearly interacting neurons," *Journal of Mathematical Biology*, vol. 4, pp. 303–321, 1977.

[7] S. Song, K. D. Miller, and L. F. Abbott, "Competitive Hebbian learning through spike-timing-dependent synaptic plasticity," *Nature Neuroscience*, vol. 3, no. 9, pp. 919–926, 2000.

[8] M. C. W. van Rossum, G. Q. Bi, and G. G. Turrigiano, "Stable Hebbian learning from spike timing-dependent plasticity," *Journal of Neuroscience*, vol. 20, no. 23, pp. 8812–8821, 2000.

[9] P. Hafliger and M. Mahowald, "Spike based normalizing Hebbian learning in an analog VLSI artificial neuron," in *Learning On Silicon* (G. Cauwenberghs and M. A. Bayoumi, eds.), pp. 131–142, Norwell, MA: Kluwer Academic, 1999.

[10] T. Lehmann and R. Woodburn, "Biologically-inspired on-chip learning in pulsed neural networks," *Analog Integrated Circuits and Signal Processing*, vol. 18, no. 2-3, pp. 117–131, 1999.

[11] A. Bofill, A. F. Murray, and D. P. Thompson, "Circuits for VLSI implementation of temporally-asymmetric Hebbian learning," in *Advances in Neural Information Processing Systems 14* (T. Dietterich, S. Becker, and Z. Ghahramani, eds.), Cambridge, MA: MIT Press, 2002.

[12] M. Mahowald, *An analog VLSI system for stereoscopic vision*. Boston: Kluwer Academic Publishers, 1994.

[13] J. Lazzaro, J. Wawrzynek, M. Mahowald, M. Sivilotti, and D. Gillespie, "Silicon auditory processors as computer peripherals," *IEEE Trans. Neural Networks*, vol. 4, no. 3, pp. 523–528, 1993.

[14] K. A. Boahen, "Point-to-point connectivity between neuromorphic chips using address events," *IEEE Trans. Circuits and Systems—II: Analog and Digital Signal Processing*, vol. 47, no. 5, pp. 416–434, 2000.

[15] C. M. Higgins and C. Koch, "Multi-chip neuromorphic motion processing," in *Proceedings 20th Anniversary Conference on Advanced Research in VLSI* (D. Wills and S. DeWeerth, eds.), (Los Alamitos, CA), pp. 309–323, IEEE Computer Society, 1999.

[16] S.-C. Liu, J. Kramer, G. Indiveri, T. Delbrück, and R. Douglas, "Orientation-selective aVLSI spiking neurons," in *Advances in Neural Information Processing Systems 14* (T. Dietterich, S. Becker, and Z. Ghahramani, eds.), Cambridge, MA: MIT Press, 2002.

[17] D. H. Goldberg, G. Cauwenberghs, and A. G. Andreou, "Probabilistic synaptic weighting in a reconfigurable network of VLSI integrate-and-fire neurons," *Neural Networks*, vol. 14, no. 6/7, pp. 781–793, 2001.

[18] C. Koch, *Biophysics of Computation: Information Processing in Single Neurons*. New York: Oxford University Press, 1999.
